# CAPACITY FOR PATTERNS AND SEQUENCES IN KANERVA'S SDM AS COMPARED TO OTHER ASSOCIATIVE MEMORY MODELS

James D. Keeler

*Chemistry Department, Stanford University, Stanford, CA 94305
and RIACS, NASA-AMES 230-5 Moffett Field, CA 94035.
e-mail: jdk@hydra.riacs.edu*

## ABSTRACT

*The information capacity of Kanerva's Sparse, Distributed Memory (SDM) and Hopfield-type neural networks is investigated. Under the approximations used here, it is shown that the total information stored in these systems is proportional to the number connections in the network. The proportionality constant is the same for the SDM and Hopfield-type models independent of the particular model, or the order of the model. The approximations are checked numerically. This same analysis can be used to show that the SDM can store sequences of spatiotemporal patterns, and the addition of time-delayed connections allows the retrieval of context dependent temporal patterns. A minor modification of the SDM can be used to store correlated patterns.*

## INTRODUCTION

Many different models of memory and thought have been proposed by scientists over the years. In (1943) McCulloch and Pitts proposed a simple model neuron with two states of activity (on and off) and a large number of inputs.[1] Hebb (1949) considered a network of such neurons and postulated mechanisms for changing synaptic strengths [2] to learn memories. The learning rule considered here uses the outer-product of patterns of +1s and -1s. Anderson (1977) discussed the effect of iterative feedback in such a system.[3] Hopfield (1982) showed that for symmetric connections,[4] the dynamics of such a network is governed by an energy function that is analogous to the energy function of a spin glass.[5] Numerous investigations have been carried out on similar models.[6-8]

Several limitations of these binary interaction, outer-product models have been pointed out. For example, the number of patterns that can be stored in the system (its capacity) is limited to a fraction of the length of the pattern vectors. Also, these models are not very successful at storing correlated patterns or temporal sequences.

Other models have been proposed to overcome these limitations. For example, one can allow higher-order interactions among the neurons.[9,10] In the following, I focus on a model developed by Kanerva (1984) called the Sparse, Distributed Memory (SDM) model.[11] The SDM can be viewed as a three layer network that uses an outer-product learning between the second and third layer. As discussed below, the SDM is more versatile than the above mentioned networks because the number of stored patterns can increased independent of the length of the pattern, and the SDM can be used to store spatiotemporal patterns with context retrieval, and store correlated patterns.

The capacity limitations of outer-product models can be alleviated by using higher-order interaction models or the SDM, but a price must be paid for this added capacity in terms of an increase in the number of connections. How much information is gained per connection? It is shown in the following that the total information stored in each system is proportional to the number of connections in the network, and that the proportionality constant is independent of the particular model or the order of the model. This result also holds if the connections are limited to one bit of precision (clipped weights). The analysis presented here requires certain simplifying assumptions. The approximate results are compared numerically to an exact calculation developed by Chou.[12]

## SIMPLE OUTER-PRODUCT NEURAL NETWORK MODEL

As an example or a simple first-order neural network model, I consider in detail the model developed by Hopfield.[4] This model will be used to introduce the mathematics and the concepts that will be generalized for the analysis of the SDM. The "neurons" are simple two-state

threshold devices: The state of the $i^{th}$ neuron, $u_i$, is either either +1 (on), or -1 (off). Consider a set of $n$ such neurons with net input (local field), $h_i$, to the $i^{th}$ neuron given by

$$h_i = \sum_{j}^{n} T_{ij} u_j, \qquad (1)$$

where $T_{ij}$ represents the interaction strength between the $i^{th}$ neuron and the $j^{th}$. The state of each neuron is updated asynchronously (at random) according to the rule

$$u_i \leftarrow g(h_i), \qquad (2)$$

where the function $g$ is a simple threshold function $g(x) = sign(x)$.

Suppose we are given $M$ randomly chosen patterns (strings of length $n$ of ±1s) which we wish to store in this system. Denote these $M$ memory patterns as pattern vectors: $\mathbf{p}^{\alpha} = (p_1^{\alpha}, p_2^{\alpha}, \ldots, p_n^{\alpha})$, $\alpha = 1, 2, 3, \ldots, M$. For example, $\mathbf{p}^1$ might look like $(+1, -1, +1, -1, -1, \ldots, +1)$. One method of storing these patterns is the outer-product (Hebbian) learning rule: Start with $T \equiv 0$, and accumulate the outer-products of the pattern vectors. The resulting connection matrix is given by

$$T_{ij} = \sum_{\alpha=1}^{M} p_i^{\alpha} p_j^{\alpha}, \quad T_{ii} = 0. \qquad (3)$$

The system described above is a dynamical system with attracting fixed points. To obtain an approximate upper bound on the total information stored in this network, we sidestep the issue of the basins of attraction, and we check to see if each of the patterns stored by Eq. (3) is actually a fixed point of (2). Suppose we are given one of the patterns, $\mathbf{p}^{\beta}$, say, as the initial configuration of the neurons. I will show that $\mathbf{p}^{\beta}$ is expected to be a fixed point of Eq. (2). After inserting (3) for $T$ into (1), the net input to the $i^{th}$ neuron becomes

$$h_i = \sum_{\alpha=1}^{M} p_i^{\alpha} [\sum_{j}^{n} p_j^{\alpha} p_j^{\beta}]. \qquad (4)$$

The important term in the sum on $\alpha$ is the one for which $\alpha = \beta$. This term represents the "signal" between the input $\mathbf{p}^{\beta}$ and the desired output. The rest of the sum represents "noise" resulting from crosstalk with all of the other stored patterns. The expression for the net input becomes $h_i = signal_i + noise_i$ where

$$signal_i = p_i^{\beta} [\sum_{j}^{n} p_j^{\beta} p_j^{\beta}], \qquad (5)$$

$$noise_i = \sum_{\alpha \neq \beta}^{M} p_i^{\alpha} [\sum_{j}^{n} p_j^{\alpha} p_j^{\beta}]. \qquad (6)$$

Summing on all of the $j_k$ in (6) yields $signal_i = (n-1)p_i^{\beta}$. Since $n$ is positive, the sign of the signal term and $p_i^{\beta}$ will be the same. Thus, if the noise term were exactly zero, the signal would give the same sign as $p_i^{\beta}$ with a magnitude of $\approx n^d$, and $\mathbf{p}^{\beta}$ would be a fixed point of (2). Moreover, patterns close to $\mathbf{p}^{\beta}$ would give nearly the same signal, so that $\mathbf{p}^{\beta}$ should be an attracting fixed point.

For randomly chosen patterns, $<noise> = 0$, where $<\ >$ indicates statistical expectation, and its variance will be $\sigma^2 = (n-1)^d (M-1)$. The probability that there will be an error on recall of $p_i^{\beta}$ is given by the probability that the noise is greater than the signal. For $n$ large, the noise distribution is approximately gaussian, and the probability that there is an error in the $i^{th}$ bit is

$$P_e = \frac{1}{\sqrt{2\pi}\sigma} \int_{|signal|}^{\infty} e^{-x^2/2\sigma^2} dx. \qquad (7)$$

## INFORMATION CAPACITY

The number of patterns that can be stored in the network is known as its capacity.[13,14] However, for a fair comparison between all of the models discussed here, it is more relevant to compare the total number of bits (total information) stored in each model rather than the number of

patterns. This allows comparison of information storage in models with different lengths of the pattern vectors. If we view the memory model as a black box which receives input bit strings and outputs them with some small probability of error in each bit, then the definition of bit-capacity used here is exactly the definition of channel capacity used by Shannon.[15]

Define the *bit-capacity* as the number of bits that can be stored in a network with fixed probability of getting an error in a recalled bit, *i.e.* $p_e = constant$ in (10). Explicitly, the bit-capacity is given by [16]

$$B = bit\ capacity = nM\eta, \tag{8}$$

where $\eta = (1 + p_e\log_2 p_e + (1-p_e)\log_2(1-p_e))$. Note that $\eta \approx 1$ for $p_e \approx 0$. Setting $p_e$ to a constant is tantamount to keeping the signal-to-noise ratio (fidelity) constant, where the fidelity, $R$, is given by $R = |signal|/\sigma$. Explicitly, the relation between (constant) $p_e$ and $R$, is just $R = \Phi^{-1}(1 - p_e)$, where

$$\Phi(R) = (1/2\pi)^{1/2}\int_{-\infty}^{R} e^{-t^2/2} dt. \tag{9}$$

Hence, the bit-capacity of these networks can be investigated by examining the fidelity of the models as a function of $n$, $M$, and $R$. From (8) and (9) the fidelity of the Hopfield model is is $R^2 = n/(n(M-1))^{1/2}$ $(n \gg 1)$. Solving for $M$ in terms of (fixed) $R$ and $\eta$, the bit-capacity becomes $B = \eta[(n^2/R^2)+n]$.

The results above can be generalized to models with $d^{th}$ order interactions.[17,18] The resulting expression for the bit-capacity for $d^{th}$ order interaction models is just

$$B = \eta[\frac{n^{d+1}}{R^2}+n]. \tag{10}$$

Hence, we see that the number of bits stored in the system increases with the order $d$. However, to store these bits, one must pay a price by including more connections in the connection tensor. To demonstrate the relationship between the number of connections and the information stored, define the *information capacity*, $\gamma$, to be the total information stored in the network divided by the number of bits in the connection tensor (note that this is different than the definition used by Abu-Mostafa *et al.*).[19] Thus $\gamma$ is just the bit-capacity divided by the number of bits in the tensor $T$, and represents the efficiency with which information is stored in the network. Since $T$ has $n^{d+1}$ elements, the information capacity is found to be

$$\gamma = \frac{\eta}{R^2 b}, \tag{11}$$

where $b$ is the number of bits of precision used per tensor element ($b \geq \log_2 M$ for no clipping of the weights). For large $n$, the information stored per neuronal connection is $\gamma = \eta/R^2 b$, independent of the order of the model (compare this result to that of Peretto, *et al.*).[20] To illustrate this point, suppose one decides that the maximum allowed probability of getting an error in a recalled bit is $p_e = 1/1000$, then this would fix the minimum value of R at 3.1. Thus, to store 10,000 bits with a probability of getting an error of a recalled bit of 0.001, equation (15) states that it would take $\approx 96,000b$ bits, independent of the order of the model, or $\approx 0.1n$ patterns can be stored with probability 1/1000 of getting an error in a recalled bit.

## KANERVA'S SDM

Now we focus our attention on Kanerva's Sparse, Distributed Memory model (SDM).[11] The SDM can be viewed as a 3-layer network with the middle layer playing the role of hidden units. To get an autoassociative network, the output layer can be fed back into the input layer, effectively making this a two layer network. The first layer of the SDM is a layer of $n$, $\pm 1$ input units (the input address, **a**), the middle layer is a layer of $m$, hidden units, **s**, and the third layer consists of the $n$ $\pm 1$ output units (the data, **d**). The connections between the input units and the hidden units are random weights of $\pm 1$ and are given by the $m \times n$ matrix $A$. The connections between the hidden units and the output units are given by the $n \times m$ connection matrix $C$, and these matrix elements are modified by an outer-product learning rule ($C$ is analogous to the matrix $T$ of the Hopfield model).

Given an input pattern **a**, the hidden unit activations are determined by

$$s = \theta_r (A\,a),$$ (12)

where $\theta_r$ is the Hamming-distance threshold function: The $k^{th}$ element is 1 if the input **a** is at most $r$ Hamming units away from the $k^{th}$ row in $A$, and 0 if it is further than $r$ units away, *i.e.*,

$$\theta_r(\mathbf{x})_i = \begin{cases} 1 & \text{if } \frac{1}{2}(n - x_i) \le r \\ 0 & \text{if } \frac{1}{2}(n - x_i) > r \, . \end{cases}$$ (13)

The hidden-units vector, or *select* vector, **s**, is mostly 0s with an average of $\delta m$ 1s, where $\delta$ is some small number dependent on $r$; $\delta \ll 1$. Hence, **s** represents a large, sparsely coded vector of 0s and $\delta 1$s representing the input address. The net input, **h**, to the final layer can be simply expressed as the product of $C$ with **s**:

$$\mathbf{h} = C\,\mathbf{s}.$$ (14)

Finally, the output data is given by $\mathbf{d} = \mathbf{g}(\mathbf{h})$, where $g_i(h_i) = sign(h_i)$.

To store the $M$ patterns, $\mathbf{p}^1, \mathbf{p}^2, \cdots \mathbf{p}^M$, form the outer-product of these pattern vectors and their corresponding select vectors,

$$C = \sum_{\alpha=1}^{M} \mathbf{p}^\alpha \mathbf{s}^{\alpha T}.$$ (15)

where $T$ denotes the transpose of the vector, and where each select vector is formed by the corresponding address, $\mathbf{s}^\alpha = \theta_r (A\,\mathbf{p}^\alpha)$. The storage algorithm (15) is an outer-product learning rule similar to (3).

Suppose that the $M$ patterns $(\mathbf{p}^1, \mathbf{p}^2, \cdots \mathbf{p}^M)$ have been stored according to (15). Following the analysis presented for the Hopfield model, I show that if the system is presented with $\mathbf{p}^\beta$ as input, the output will be $\mathbf{p}^\beta$, (*i.e.* $\mathbf{p}^\beta$ is a fixed point). Setting $\mathbf{a} = \mathbf{p}^\beta$ in (16) and separating terms as before, the net input (18) becomes

$$\mathbf{h} = \mathbf{d}^\beta (\mathbf{s}^\beta \cdot \mathbf{s}^\beta) + \sum_{\alpha \ne \beta}^{M} \mathbf{p}^\alpha (\mathbf{s}^\alpha \cdot \mathbf{s}^\beta).$$ (16)

where the first term represents the signal and the second is the noise. Recall that the select vectors have an average of $\delta m$ 1s and the remainder 0s, so that the expected value of the signal is $\delta m\, \mathbf{s}^\beta$.

Assuming that the addresses and data are randomly chosen, the expected value of the noise is zero. To evaluate the fidelity, I make certain approximations. First, I assume that the select vectors are independent of each other. Second, I assume that the variance of the signal alone is zero or small compared to the variance of noise term alone. The first assumption will be valid for $m\delta^2 \ll 1$, and the second assumption will be valid for $M\delta \gg 1$. With these assumptions, we can easily calculate the variance of the noise term, because each of the select vectors are i.i.d. vectors of length $m$ with mostly 0s and $\approx \delta m$ 1s. With these assumptions, the fidelity is given by

$$R^2 = \frac{m}{[(M-1)(1 + \delta^2 m (1 - 1/m))]}.$$ (17)

In the limit of large $m$, with $\delta m \approx constant$, the number of stored bits scales as

$$B = \eta [\frac{mn}{R^2 (1 + \delta^2 m)} + n].$$ (18)

If we divide this by the number of elements in $C$, we find the information capacity, $\gamma = \eta/R^2 b$, just as before, so the information capacity is the same for the two models. (If we divide the bit capacity by the number of elements in $C$ and $A$ then we get $\gamma = \eta/R^2(b+1)$, which is about the same for large $M$.)

A few comments before we continue. First, it should be pointed out that the assumption made by Kanerva[11] and Keeler[17,18] that the variance of the signal term is much less than that of the noise is not valid over the entire range. If we took this into account, then the magnitude of the denominator would be increased by the variance of the signal term. Further, if we read at a distance $l$ away from the write address, then it is easy to see that the signal changes to be $m\,\delta(l)$, where $\delta(l)$ the overlap of two spheres of radius $r$ length $l$ apart in the binomial space $n$

$(\delta \equiv \delta(0))$. The fidelity for reading at a distance $l$ away from the write address is

$$R^2 = \frac{m^2\delta^2(l)}{m\,\delta(l)(1-\delta(l)) + (M-1)m\,\delta^2 + (M-1)\delta^4 m^2(1-1/m)}, \qquad (19)$$

Compare this to the formula derived by Chou,[12] for the exact signal-to-noise ratio:

$$R^2 = \frac{m^2\delta^2(l)}{m\,\delta(l)(1-\delta(l)) + (M-1)m\,\mu_{n,r} + (M-1)\sigma^2_{n,r}m^2(1-1/m))}, \qquad (20)$$

where $\mu_{n,r}$ is the average overlap of the spheres of radius $r$ binomially distributed with parameters $(n,1/2)$ and $\sigma^2$ is the square of this overlap. The difference in these two formulas lies in the denominator in the terms $\delta^2$ verses $\mu_{n,r}$ and $\delta^4$ vs. $\sigma^2_{n,r}$. The difference comes from the fact that Chou correctly calculates the overlap of the spheres without using the independence assumption.

How do these formula's differ? First of all, it is found numerically that $\delta^2$ is identical with $\mu_{n,r}$. Hence, the only difference comes from $\delta^4$ verses $\sigma^2_{n,r}$. For $m\delta^2 \ll 1$, the $\delta^4$ term is negligible compared to the other terms in the denominator. In addition, $\delta^4$ and $\sigma^2$ are approximately equal for large $n$ and $r \approx n/2$. Hence, in the limit $n \rightarrow \infty$ the two formulas agree over most of the range if $M \approx 0.1m$, $m \ll 2^n$. However, for finite $n$, the two formulas can disagree when $m\,\delta^2 \approx 1$ (see Figure 1).

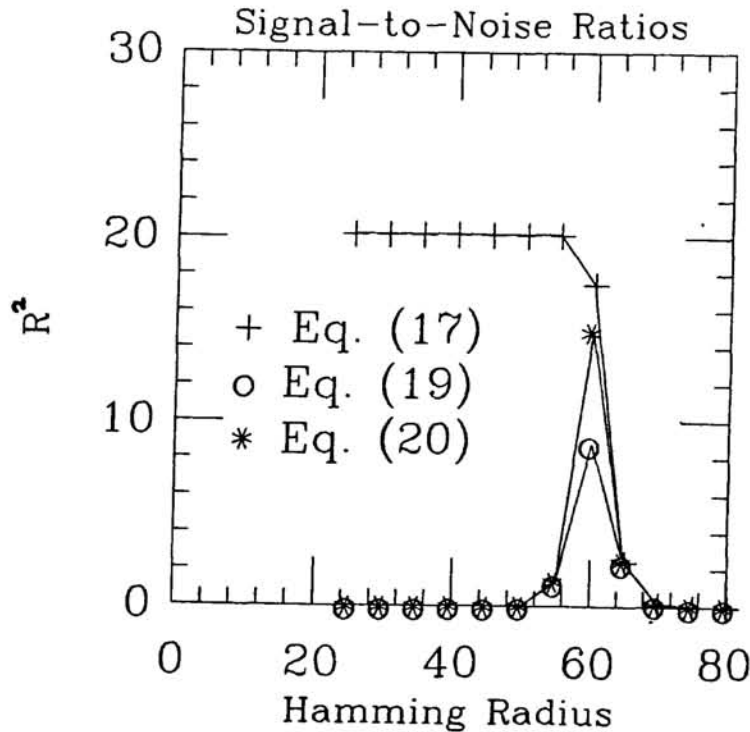

**Figure 1:** A comparison of the fidelity calculations of the SDM for typical $n$, $M$, and $m$ values. Equation (17) was derived assuming no variance of the signal term, and is shown by the + line. Equation (19) uses the approximation that all of the select vectors are independent denoted by the $o$ line. Equation (20) (*'s) is the exact derivation done by Chou[12]. The values used here were $n = 150$, $m = 2000$, $M = 100$.

Equation (20) suggests that there is a best read-write Hamming radius for the SDM. By setting $l = 0$ in (19) and by setting $\dfrac{dR^2}{d\delta} = 0$, we get an approximate expression for the best Hamming radius: $\delta_{best} \approx (2Mm)^{-1/3}$. This trend is qualitatively shown in Figure 2.

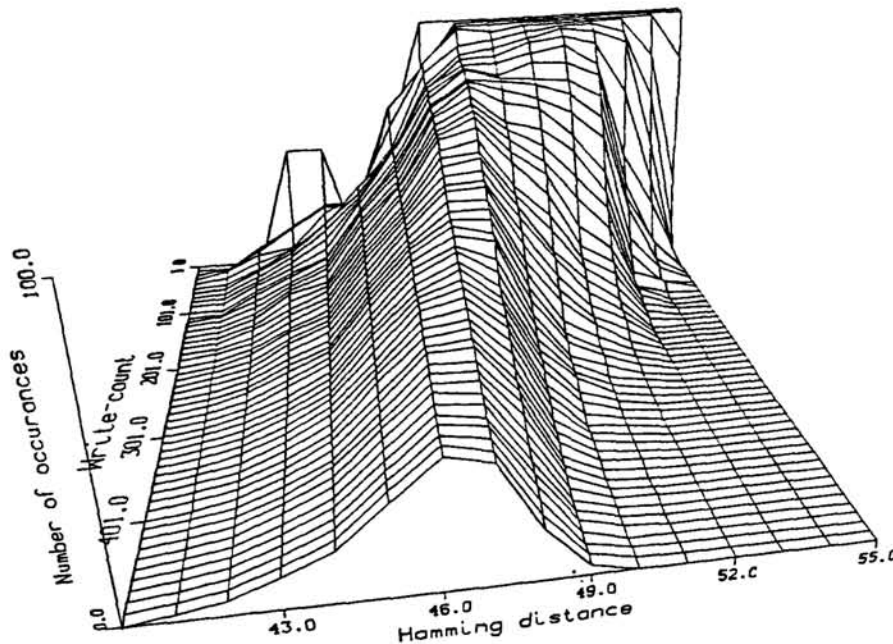

**Figure 2:** Numerical investigation of the capacity of the SDM. The vertical axis is the percent of recovered patterns with no errors. The x-axis (left to right) is the Hamming distance used for reading and writing. The y-axis (back to forward) is the number of patterns that were written into the memory. For this investigation, $n = 128$, $m = 1024$, and $M$ ranges from 1 to 501. Note the similarity of a cross-section of this graph at constant $M$ with Figure 1. This calculation was performed by David Cohn at RIACS, NASA-Ames.

Figure 1 indicates that the formula (17) that neglected the variance of the signal term is incorrect over much of the range. However, a variant of the SDM is to constrain the number of selected locations to be constant; circuitry for doing this is easily built.[21] The variance of the signal term would be zero in that case, and the approximate expression for the fidelity is given by Eq. (17). There are certain problems where it would be better to keep $\delta = constant$, as in the case of correlated patterns (see below).

The above analysis was done assuming that the elements (weights) in the outer-product matrix are not clipped *i.e.* that there are enough bits to store the largest value of any matrix element. It is interesting to consider what happens if we allow these values to be represented by only a few bits. If we consider the case case $b = 1$, *i.e.* the weights are clipped at one bit, it is easy to show[17] that $\gamma \approx 2\eta/\pi R^2$ for the $d^{th}$ order models and for the SDM, which yields $\gamma = 0.07$ for reasonable $R$, (this is substantially less than Willshaw's 0.69).

## SEQUENCES

In an autoassociative memory, the system relaxes to one of the stored patterns and stays fixed in time until a new input is presented. However, there are many problems where the recalled patterns must change sequentially in time. For example, a song can be remembered as a string of notes played in the correct sequence; cyclic patterns of muscle contractions are essential for walking, riding a bicycle, or dribbling a basketball. As a first step we consider the very simplistic sequence production as put forth by Hopfield (1982) and Kanerva (1984).

Suppose that we wished to store a sequence of patterns in the SDM. Let the pattern vectors be given by $(\mathbf{p}^1, \mathbf{p}^2, \ldots, \mathbf{p}^M)$. This sequence of patterns could be stored by having each pattern point to the next pattern in the sequence. Thus, for the SDM, the patterns would be stored as input-output pairs $(\mathbf{a}^\alpha, \mathbf{d}^\alpha)$, where $\mathbf{a}^\alpha = \mathbf{p}^\alpha$ and $\mathbf{d}^\alpha = \mathbf{p}^{\alpha+1}$ for $\alpha = 1,2,3,\ldots,M-1$. Convergence to this sequence works as follows: If the SDM is presented with an address that is close to $\mathbf{p}^1$ the read data will be close to $\mathbf{p}^2$. Iterating the system with $\mathbf{p}^2$ as the new input address, the read data will be even closer to $\mathbf{p}^3$. As this iterative process continues, the read data will converge to the stored sequence, with the next pattern in the sequence being presented at each time step.

The convergence statistics are essentially the same for sequential patterns as that shown above for autoassociative patterns. Presented with $\mathbf{p}^\alpha$ as an input address, the signal for the stored sequence is found as before

$$\langle \text{signal} \rangle = \delta m\, \mathbf{p}^{\alpha+1}. \tag{21}$$

Thus, given $\mathbf{p}^\alpha$, the read data is expected to be $\mathbf{p}^{\alpha+1}$. Assuming that the patterns in the sequence are randomly chosen, the mean value of the noise is zero, with variance

$$\langle \sigma^2 \rangle = (M-1)\delta^2 m\, (1+\delta^2(m-1)). \tag{22}$$

Hence, the length of a sequence that can be stored in the SDM increases linearly with $m$ for large $m$.

Attempting to store sequences like this in the Hopfield model is not very successful due to the asynchronous updating use in the Hopfield model. A synchronously updated outer-product model (for example [6]) would work just as described for the SDM, but it would still be limited to storing fraction of the word size as the maximum sequence length.

Another method for storing sequences in Hopfield-like networks has been proposed independently by Kleinfeld[22] and Sompolinsky and Kanter.[23] These models relieve the problem created by asynchronous updating by using a time-delayed sequential term. This time-delay storage algorithm has different dynamics than the synchronous SDM model. In the time-delay algorithm, the system allows time for the units to relax to the first pattern before proceeding on to the next pattern, whereas in the synchronous algorithms, the sequence is recalled imprecisely from imprecise input for the first few iterations and then correctly after that. In other words, convergence to the sequence takes place "on the fly" in the synchronous models — the system does not wait to zero in on the first pattern before proceeding on to recover the following patterns. This allows the synchronous algorithms to proceed $k$ times as fast as the asynchronous time-delay algorithms with half as many (variable) matrix elements. This difference should be able to be detected in biological systems.

## TIME DELAYS AND HYSTERESIS: FOLDS

The above scenario for storing sequences is inadequate to explain speech recognition or pattern generation. For example, the above algorithm cannot store sequences of the form $ABAC$, or overlapping sequences. In Kanerva's original work, he included the concept of time delays as a general way of storing sequences with hysteresis. The problem addressed by this is the following: Suppose we wish to store two sequences of patterns that overlap. For example, the two pattern sequences $(a,b,c,d,e,f,\ldots)$ and $(x,y,z,d,w,v,\ldots)$ overlap at the pattern $\mathbf{d}$. If the system only has knowledge of the present state, then when given the input $\mathbf{d}$, it cannot decide whether to output $\mathbf{w}$ or $\mathbf{e}$. To store two such sequences, the system must have some knowledge of the immediate past. Kanerva incorporates this idea into the SDM by using "folds." A system with $F+1$ folds has a time history of $F$ past states. These $F$ states may be over the past $F$ time steps or they may go even further back in time, skipping some time steps. The algorithm for reading from the SDM with folds becomes

$$\mathbf{d}(t+1) = \mathbf{g}(C^0 \cdot \mathbf{s}(t) + C^1 \cdot \mathbf{s}(t-\tau_1) + \cdots + C^F \cdot \mathbf{s}(t-\tau_F)), \tag{23}$$

where $s(t-\tau_\beta) = \theta_r(A\,a(t-\tau_\beta))$. To store the $Q$ pattern sequences $(\mathbf{p}_1^1, \mathbf{p}_1^2, \ldots, \mathbf{p}_1^{M_1})$, $(\mathbf{p}_2^1, \mathbf{p}_2^2, \ldots, \mathbf{p}_2^{M_2}), \ldots (\mathbf{p}_Q^1, \mathbf{p}_Q^2, \ldots, \mathbf{p}_Q^{M_Q})$, construct the matrix of the $\beta^{th}$ fold as follows:

$$C^\beta = w_\beta \sum_{\alpha=1}^{Q} \sum_{\tau=1}^{M_\beta} \mathbf{p}_\alpha^{\tau+1} \times \mathbf{s}_\alpha^{\tau-\tau_\beta}, \tag{24}$$

where any vector with a superscript less than 1 is taken to be zero, $\mathbf{s}_\alpha^{\tau-\tau_x} = \theta_r(A\,\mathbf{p}_\alpha^{\tau-\tau_x})$, and $w_\beta$ is a weighting factor that would normally decrease with increasing $\beta$.

Why do these folds work? Suppose that the system is presented with the pattern sequence $(\mathbf{p}_1^1, \mathbf{p}_1^2, \ldots, \mathbf{p}_1^{M_1})$, with each pattern presented sequentially as input until the $\tau_F$ time step. For simplicity, assume that $w_\beta = 1$ for all $\beta$. Each term in Eq. (39) will contribute a signal similar to the signal for the single-fold system. Thus, on the $\tau^{th}$ time step, the signal term coming from Eq. (39) is $\langle \text{signal}(t+1)\rangle = F\,\delta m\,\mathbf{p}_1^{\tau+1}$. The signal will have this value until the end of the pattern sequence is reached. The mean of the noise terms is zero, with variance $\langle \text{noise}^2 \rangle = F(M-1)\delta^2 m(1+\delta^2(m-1))$. Hence, the signal-to-noise ratio is $\sqrt{F}$ times as strong as it is for the SDM without folds.

Suppose further that the second stored pattern sequence happens to match the first stored sequence at $t = \tau$. The signal term would then be

$$\text{signal}(t+1) = F\,\delta m\,\mathbf{p}_1^{\tau+1} + \delta m\,\mathbf{p}_2^{\tau+1}. \tag{25}$$

With no history of the past ($F = 1$) the signal is split between $\mathbf{p}_1^{\tau+1}$ and $\mathbf{p}_2^{\eta+1}$, and the output is ambiguous. However, for $F>1$, the signal for the first pattern sequence dominates and allows retrieval of the remainder of the correct sequence. This formulation allows context to aid in the retrieval of stored sequences, and can differentiate between overlapping sequences by using time delays.

The above formulation is still too simplistic in terms of being able to do real recognition problems such as speech recognition. First, the above algorithm can only recall sequences at a fixed time rate, whereas speech recognition occurs at widely varying rates. Second, the above algorithm does not allow for deletions in the incoming data. For example "seqnce" can be recognized as "sequence" even though some letters are missing. Third, as pointed out by Lashley[24] speech processing relies on hierarchical structures.

Although Kanerva's original algorithm is too simplistic, a straightforward modification allows retrieval at different rates with deletions. To achieve this, we can add on the time-delay terms with weights which are smeared out in time. Kanerva's (1984) formulation can thus be viewed as a discrete-time formulation of that put forth by Hopfield and Tank, (1987).[25] Explicitly we could write

$$\mathbf{h} = \sum_{\beta=1}^{F} \sum_{k=\beta-F}^{\beta} W_{\beta k} C^\beta s(t-\tau_{\beta-k}), \tag{26}$$

where the coefficients $W_{\beta k}$ are a discrete approximation to a smooth function which spreads the delayed signal out over time. As a further step, we could modify these weights dynamically to optimize the signal coming out. The time-delay patterns could also be placed in a hierarchical structure as in the matched filter avalanche structure put forth by Grossberg et al. (1986).[26]

## CORRELATED PATTERNS

In the above associative memories, all of the patterns were taken to be randomly chosen, uniformly distributed binary vectors of length $n$. However, there are many applications where the set of input patterns is not uniformly distributed; the input patterns are correlated. In mathematical terms, the set $\kappa$ of input patterns would not be uniformly distributed over the entire space of $2^n$ possible patterns. Let the probability distribution function for the Hamming distance between two randomly chosen vectors $\mathbf{p}^\alpha$ and $\mathbf{p}^\beta$ from the distribution $\kappa$ be given by the function $\rho(d(\mathbf{p}^\alpha-\mathbf{p}^\beta))$, where $d(\mathbf{x}-\mathbf{y})$ is the Hamming distance between $\mathbf{x}$ and $\mathbf{y}$.

The SDM can be generalized from Kanerva's original formulation so that correlated input patterns can be associated with output patterns. For the moment, assume that the distribution set $\kappa$ and the probability density function $\rho(x)$ are known a priori. Instead of constructing the rows of the matrix $A$ from the entire space of $2^n$ patterns, construct the rows of $A$ from the distribution $\kappa$. Adjust the Hamming distance $r$ so that $\zeta = \delta m = constant$ number of locations are selected.

In other words, adjust $r$ so that the value of $\delta$ is the same as given above, where $\delta$ is determined by

$$\delta = \frac{\int_0^r \rho(x)\,dx}{2^n}. \tag{27}$$

This implies that $r$ would have to be adjusted dynamically. This could be done, for example, by a feedback loop. Circuitry for doing this is easily built,[21] and a similar structure appears in the Golgi cells in the Cerebellum.[27].

Using the same distribution for the rows of $A$ as the distribution of the patterns in $\kappa$, and using (27) to specify the choice of $r$, all of the above analysis is applicable (assuming randomly chosen output patterns). If the outputs do not have equal 1s and −1s the mean of the noise is not 0. However, if the distribution of outputs is also known, the system can still be made to work by storing $1/p_+$ and $1/p_-$ for 1s and -1s respectively, where $p_\pm$ is the probability of getting a 1 or a −1 respectively. Using this storage algorithm, all of the above formulas hold, (as long as the distribution is smooth enough and not extremely dense). The SDM will be able to recover data stored with correlated inputs with a fidelity given by Equation (17).

What if the distribution function $\kappa$ is not known *a priori*? In that case, we would need to have the matrix $A$ learn the distribution $\rho(\mathbf{x})$. There are many ways to build $A$ to mimic $\rho$. One such way is to start with a random $A$ matrix and modify the entries of $\delta$ randomly chosen rows of $A$ at each step according to the statistics of the most recent input patterns. Another method is to use competitive learning[28-30] to achieve the proper distribution of $A_t$.

The competitive learning algorithm is a method for adjusting the weights $A_{ij}$ between the first and second layer to match this probability density function, $\rho(\mathbf{x})$. The $i^{th}$ row of the address matrix $A$ can be viewd as a vector $A_i$. The competitive learning algorithm holds a competition between these vectors, and a few vectors that are the closest (within the Hamming sphere $r$) to the input pattern $\mathbf{x}$ are the winners. Each of these winners are then modified slightly in the direction of $\mathbf{x}$. For large enough $m$, this algorithm almost always converges to a distribution of the $A_i$ that is the same as $\rho(\mathbf{x})$.[xxx] The updating equation for the selected addresses is just

$$\mathbf{A}_i^{new} = \mathbf{A}_i^{old} - \lambda(\mathbf{A}_i^{old} - \mathbf{x}) \tag{28}$$

Note for $\lambda = 1$, this reduces to the so-called unary representation of Baum *et al.*[31] Which gives the maximum efficiency in terms of capacity.

## DISCUSSION

The above analysis said nothing about the basins of attraction of these memory states. A measure of the performance of a content addressable memory should also say something about the average radius of convergence of the basin of attraction. The basins are in general quite complicated[32] and have been investigated numerically for the unclipped models and values of $n$ and $m$ ranging in the 100s.[21] The basins of attraction for the SDM and the $d=1$ model are very similar in their characteristics and their average radius of convergence. However, the above results give an upper bound on the capacity by looking at the fixed points of the system (if there is no fixed point, there is no basin).

In summary, the above arguments show that the total information stored in outer-product neural networks is a constant times the number of connections between the neurons. This constant is independent of the order of the model and is the same ($\eta/R^2 b$) for the SDM as well as higher-order Hopfield-type networks. The advantage of going to an architecture like the SDM is that the number of patterns that can be stored in the network is independent of the size of the pattern, whereas the number of stored patterns is limited to a fraction of the word size for the Willshaw or Hopfield architecture. The point of the above analysis is that the efficiency of the SDM in terms of information stored per bit is the same as for Hopfield-type models.

It was also demonstrated how sequences of patterns can be stored in the SDM, and how time delays can be used to recover contextual information. A minor modification of the SDM could be used to recover time sequences at slightly different rates of presentation. Moreover, another minor modification allows the storage of correlated patterns in the SDM. With these modifications, the SDM presents a versatile and efficient tool for investigating properties of associative memory.

**Acknowledgements:** Discussions with John Hopfield and Pentti Kanerva are gratefully acknowledged. This work was supported by DARPA contract # 86-A227500-000.

**REFERENCES**

[1] McCulloch, W. S. & Pitts, W. (1943), *Bull. Math. Biophys.* **5**, 115-133.

[2] Hebb, D. O. (1949) *The Organization of Behavior.* John Wiley, New York.

[3] Anderson, J. A., Solverstein, J. W., Ritz, S. A. & Jones, R. S. (1977) *Psych. Rev.,* **84**, 412-451.

[4] Hopfield, J. J. (1982) *Proc. Natn'l. Acad. Sci. USA* **79** 2554-2558.

[5] Kirkpatrick, S. & Sherrington, D. (1978) *Phys Rev.* **17** 4384-4405.

[6] Little, W. A. & Shaw, G. L.(1978)*Math. Biosci.* **39**, 281-289.

[7] Nakano, K. (1972), Association - A model of associative memory, *IEEE Trans. Sys. Man Cyber.* **2**,

[8] Willshaw, D. J., Buneman, O. P. & Longuet-Higgins, H. C., (1969) *Nature*, **222** 960-962.

[9] Lee, Y. C.; Doolen, G.; Chen, H. H.; Sun, G. Z.; Maxwell, T.; Lee, H. Y.; & Giles, L. (1985) *Physica* , **22D**, 276-306.

[10] Baldi, P., and Venkatesh, S. S., (1987) Phys. Rev. Lett. **58**, 913-916.

[11] Kanerva, P. (1984) *Self-propagating Search: A Unified Theory of Memory,* Stanford University Ph.D. Thesis, and Bradford Books (MIT Press). In press (1987 est).

[12] Chou, P. A., *The capacity of Kanerva's Associative Memory* these proceedings.

[13] McEliece, R. J., Posner, E. C., Rodemich, E. R., & Venkatesh, S. S. (1986), *IEEE Trans. on Information Theory.*

[14] Amit, D. J., Gutfreund, H. & Sompolinsky, H. (1985) *Phys. Rev. Lett.* **55**, 1530-1533.

[15] Shannon, C. E., (1948), *Bell Syst. Tech. J.*, **27**, 379,623 (Reprinted in Shannon and Weaver 1949) .

[16] Kleinfeld, D. & Pendergraft, D. B., (1987) Biophys. J. **51**, 47-53.

[17] Keeler, J. D. (1986), *Comparison of Sparsely Distributed Memory and Hopfield-type Neural Network Models,* RIACS Technical Report 86-31, also submitted to *J. Cog. Sci.*

[18] Keeler, J. D. (1987) *Physics Letters* **124A**, 53-58.

[19] Abu-Mostafa, Y. & St. Jacques, (1985), IEEE Trans. on Info. Theor., **31**, 461.

[18] Keeler, J. D., *Basins of Attraction of Neural Network Models* AIP Conf. Proc. #151, Ed: John Denker, Neural Networks for Computing, Snowbird Utah, (1986).

[20] Peretto, P. & J.J. Niez, (1986) Biol. Cybern., **54**. 53-63.

[21] Keeler, J. D., Ph. D. Dissertation. *Collective phenomena of coupled lattice maps: Reaction-diffusion systems and neural networks.* Department of Physics, University of California, San Diego, (1987).

[22] Kleinfeld, D. (1986). *Proc. Nat. Acad. Sci.* **83** 9469-9473.

[23] Sompolinsky, H. & Kanter, I. (1986). *Physical Review Letters.*

[24] Lashley, K. S. (1951). *Cerebral Mechanisms in Behavior.* Edited by Jeffress, L. A. Wiley, New York, 112-136.

[25] Hopfield, J. J. & Tank, D. W. (1987). ICNN San Diego preprint.

[26] Grossberg, S. & Stone, G. (1986). *Psychological Review,* **93**, 46-74

[27] Marr, D. (1969). A *Journal of Phisiology,* **202**, 437-470.

[28] Grossberg, S. (1976). *Biological Cybernetics* **23**, 121-134.

[29] Kohonen, T. (1984) *Self-organization and associative memory.* Springer-Verlag, Berlin.

[30] Rumelhart, D. E. & Zipser, D. *J. Cognitive Sci.,* **9**, (1985), 75.

[31] Baum, E., Moody J., Wilczek F. (1987). Preprint for *Biological Cybernetics*